# Mapping Between Neural and Physical Activities of the Lobster Gastric Mill

Kenji Doya      Mary E.T. Boyle      Allen I. Selverston

Department of Biology
University of California, San Diego
La Jolla, CA 92093-0322

## Abstract

A computer model of the musculoskeletal system of the lobster gastric mill was constructed in order to provide a behavioral interpretation of the rhythmic patterns obtained from isolated stomatogastric ganglion. The model was based on Hill's muscle model and quasi-static approximation of the skeletal dynamics and could simulate the change of chewing patterns by the effect of neuromodulators.

## 1 THE STOMATOGASTRIC NERVOUS SYSTEM

The crustacean stomatogastric ganglion (STG) is a circuit of 30 neurons that controls rhythmic movement of the foregut. It is one of the best elucidated neural circuits. All the neurons and the synaptic connections between them are identified and the effects of neuromodulators on the oscillation patterns and neuronal characteristics have been extensively studied (Selverston and Moulins 1987, Harris-Warrick et al. 1992). However, STG's function as a controller of ingestive behavior is not fully understood in part because of our poor understanding of the controlled object: the musculoskeletal dynamics of the foregut. We constructed a mathematical model of the gastric mill, three teeth in the stomach, in order to predict motor patterns from the neural oscillation patterns which are recorded from the isolated ganglion.

The animal we used was the Californian spiny lobster (*Panulirus interruptus*), which

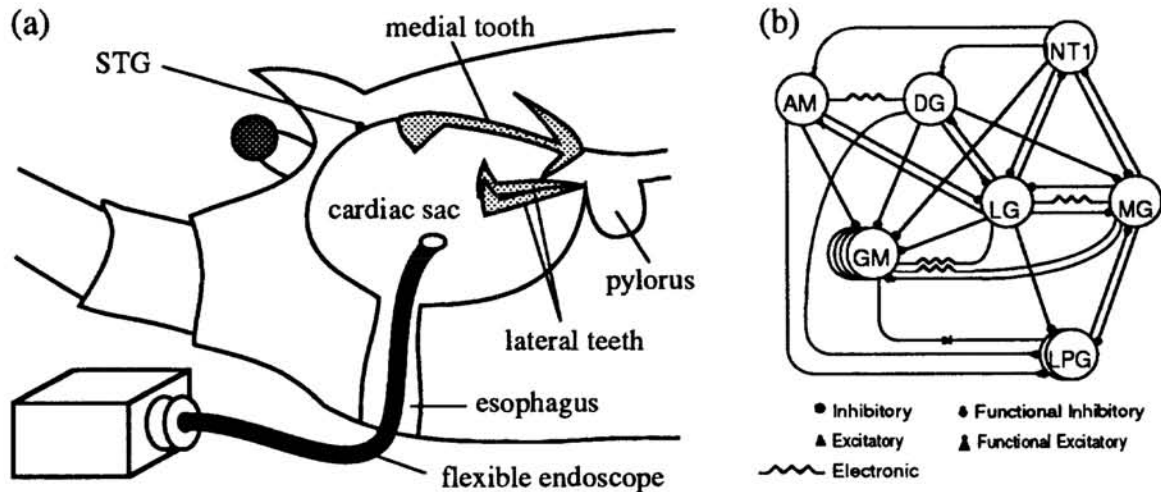

Figure 1: The lobster stomatogastric system. (a) Cross section of the foregut (objects are not to scale). (b) The gastric circuit.

is available locally. The stomatogastric nervous system controls four parts of the foregut: esophagus, cardiac sac (stomach), gastric mill, and pylorus (entrance to the intestine) (Figure 1.a). The gastric mill is composed of one medial tooth and two lateral teeth. These grind large chunks of foods (mollusks, algae, crabs, sea urchins, etc.) into smaller pieces and mix them with digestive fluids. The chewing period ranges from 5 to 10 seconds. Several different chewing patterns have been analyzed using an endoscope (Heinzel 1988a, Boyle et al. 1990). Figure 2 shows two of the typical chewing patterns: "cut and grind" and "cut and squeeze".

The STG is located in the opthalmic artery which runs from the heart to brain over the dorsal surface of the stomach. When it is taken out with two other ganglia (the esophageal ganglion and the commissural ganglion), it can still generate rhythmic motor outputs. This isolated preparation is ideal for studying the mechanism of rhythmic pattern generation by a neural circuit. From pairwise stimulus and response of the neurons, the map of synaptic connections has been established. Figure 1 (b) shows a subset of the STG circuit which controls the motion of the gastric mill. It consists of 11 neurons of 7 types. GM and DG neurons control the medial tooth and LPG, MG, and LG neurons control the lateral teeth. A question of interest is how this simple neural network is utilized to control the various movement patterns of the gastric mill, which is a fairly complex musculoskeletal system.

The oscillation pattern of the isolated ganglion can be modulated by perfusing it with of several neuromodulators, e.g. proctolin, octopamine (Heinzel and Selverston 1988), CCK (Turrigiano 1990), and pilocarpine (Elson and Selverston 1992). However, the behavioral interpretation of these different activity patterns is not well understood. The gastric mill is composed of 7 ossicles (small bones) which is loosely suspended by more than 20 muscles and connective tissues. That makes it is very difficult to intuitively estimate the effect of the change of neural firing patterns in terms of the teeth movement. Therefore we, decided to construct a quantitative model of the musculoskeletal system of the gastric mill.

(a)

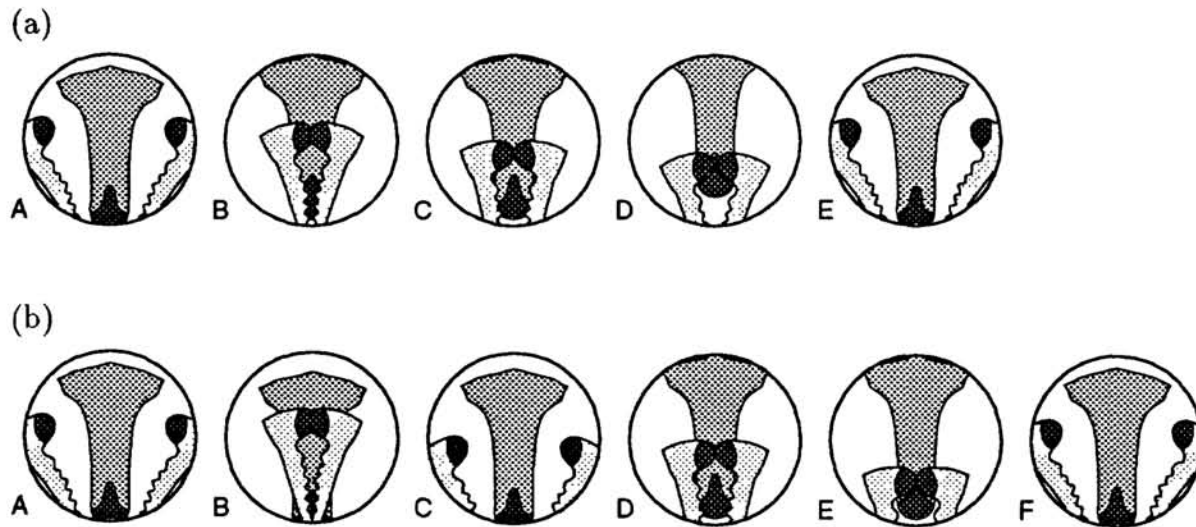

(b)

Figure 2: Typical chewing patterns of the gastric mill. (a) cut and grind. (b) cut and squeeze.

## 2   PHYSIOLOGICAL EXPERIMENTS

In order to design a model and determine its parameters, we performed anatomical and physiological experiments described below.

**Anatomical experiments:** The carapace and the skin above the stomach mill was removed to expose a dorsal view of the ossicles and the muscles which control the gastric mill. Usually, the gastric mill was quiescent without any stimuli. The positions of the ossicles and the lengths of the muscles at the resting state was measured. After the behavioral experiments mentioned below, the gastric mill was taken out and the size of the ossicles and the positions of the attachment points of the muscles were measured.

**Behavioral experiments:** With the carapace removed and the gastric mill exposed, one video camera was used to record the movement of the ossicles and the muscles. Another video camera attached to a flexible endoscope was used to record the motion of the teeth from inside the stomach. In the resting state, muscles were stimulated by a wire electrode to determine the behavioral effects. In order to induce chewing, neuromodulators such as proctolin and pilocarpine were injected into the artery in which STG is located.

**Single muscle experiments:** The gm1, the largest of the gastric mill muscles, was used to estimate the parameters of the muscle model mentioned below. It was removed without disrupting the carapace or ossicle attachment points and fixed to a tension measurement apparatus. The nerve fiber *aln* that innervates gm1 was stimulated using a suction electrode. The time course of isometric tension was recorded at different muscle lengths and stimulus frequencies. The parameters obtained from the gm1 muscle experiment were applied to other muscles by considering their relative length and thickness.

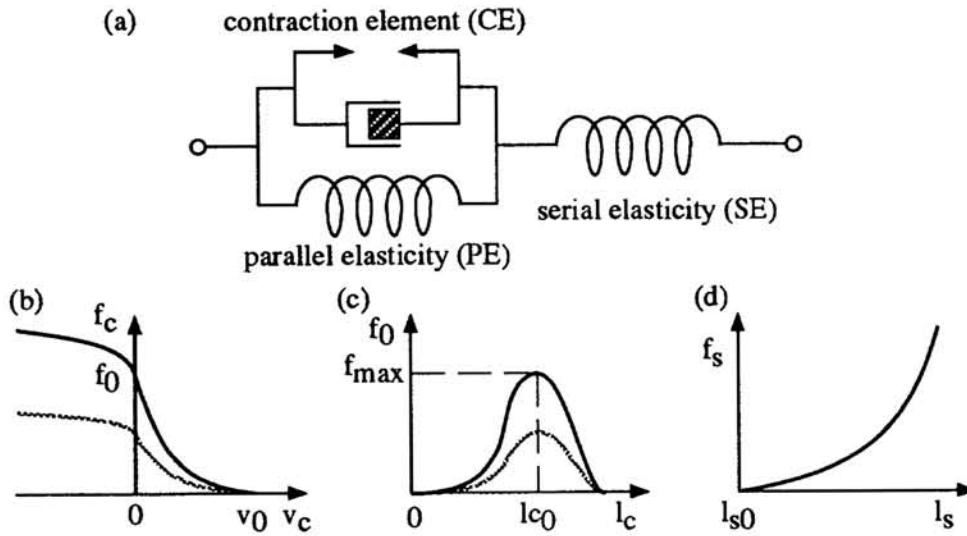

Figure 3: The Hill-based muscle model.

# 3   MODELING THE MUSCULOSKELETAL SYSTEM

## 3.1   MUSCULAR DYNAMICS

There are many ways to model muscles. In the simplest models, the tension or the length of a muscle is regarded as an instantaneous function of the spike frequency of the motor nerve. In some engineering approaches, a muscle is considered as a spring whose resting length and stiffness are modulated by the nervous input (Hogan 1984). Since these models are a linear static approximation of the nonlinear dynamical characteristics of muscles, their parameters must be changed to simulate different motor tasks (Winters90). Molecular models (Zahalak 1990), which are based on the binding mechanisms of actin and myosin fibers, can explain the widest range of muscular characteristics found in physiological experiments. However, these complex models have many parameters which are difficult to estimate.

The model we employed was a nonlinear macroscopic model based on A. V. Hill's formulation (Hill 1938, Winters 1990). The model is composed of a contractile element (CE), a serial elasticity (SE), and a parallel elasticity (PE) (Figure 3.a). This model is based on empirical data about nonlinear characteristics of muscles and its parameters can be determined by physiological experiments.

The output force $f_c$ of the CE is a function of its length $l_c$ and its contraction speed $v_c = -dl_c/dt$ (Figure 3.b)

$$\frac{f_c}{f_0} = \begin{cases} \frac{v_0 - v_c}{v_0 + v_c/\alpha} & v_c \geq 0 \text{ (contraction),} \\ 1 + \beta(1 - \frac{\beta v_0 + v_c}{\beta v_0 - v_c/\alpha}) & v_c < 0 \text{ (extension),} \end{cases} \tag{1}$$

where $f_0$ is the isometric output force (at $v_c = 0$) and $v_0$ is the maximal contraction velocity. The parameters of the $f$-$v$ curve were $\alpha = 0.25$ and $\beta = 0.3$. The isometric force $f_0$ was given as the function of CE length $l_c$ and the activation level $a(t)$ of

the muscle (Figure 3.c)

$$f_0(l_c, a(t)) = \begin{cases} f_{max} \frac{1}{1-\gamma} \left(\frac{l_c}{l_{c0}}\right)^2 \left(\frac{l_c}{l_{c0}} - \gamma\right) a(t) & 0 < l_c < \gamma, \\ 0 & \text{otherwise}, \end{cases} \quad (2)$$

where $l_{c0}$ is the resting length of the CE and $\gamma = 1.5$.

The SE was modeled as an exponential spring (Figure 3.d)

$$f_s(l_s) = \begin{cases} k_1(\exp[k_2 \frac{l_s - l_{s0}}{l_{s0}}] - 1) & l_s \geq l_{s0}, \\ 0 & l_s < l_{s0}, \end{cases} \quad (3)$$

where $f_s$ is the output force, $l_{s0}$ is the resting length, and $k_1$ and $k_2$ are stiffness parameters. The PE was supposed to have the same exponential elasticity (3).

In the simulations, the CE length $l_c$ was taken as the state variable. The total muscle length $l_m = l_c + l_s$ is given by the skeletal model and the muscle activation $a(t)$ is given by the the activation dynamics described below. The SE length is given from $l_s = l_m - l_c$ and then the output force $f_s(l_s) = f_c + f_p = f_m$ is given by (3). The contraction velocity $v_c = -\frac{dl_c}{dt}$ is derived from the inverse of (1) at $f_c = f_s(l_s) - f_p(l_c)$ and then integrated to update the CE length $l_c$.

The activation level $a(t)$ of a muscle is determined by the free calcium concentration in muscle fibers. Since we don't have enough data about the calcium dynamics in muscle cells, the activation dynamics was crudely approximated by the following equations.

$$\tau_a \frac{da(t)}{dt} = -a(t) + e(t), \quad \text{and} \quad \tau_e \frac{de(t)}{dt} = -e(t) + n(t)^2, \quad (4)$$

where $n(t)$ is the normalized firing frequency of the nerve input and $e(t)$ is the electric activity of the muscle fibers. The nonlinearity in the nervous input represents strong facilitation of the postsynaptic potential (Govind and Lingle 1987).

We incorporated seven of the gastric mill muscles: gm1, gm2, gm3a, gm3c, gm4, gm6b, and gm9a (Maynard and Dando 1974). The muscles gm1, gm2, gm3a, and gm3c are extrinsic muscles that have one end attached to the carapace and gm4, gm6b, and gm9a are intrinsic muscles both ends of which are attached of the ossicles. Three connective tissues were also incorporated and regarded as muscles without contraction elements. See Figure 4 for the attachment of these muscles and tissues to the ossicles.

## 3.2 SKELETAL DYNAMICS

The medial tooth was modeled as three rigid pieces $P_1$, $P_2$ and $P_3$. $P_1$ is the base of the medial tooth. $P_2$ is the main body of the medial tooth. $P_3$ forms the cusp and the V-shaped lever on the dorsal side. The lateral tooth was modeled as two rigid pieces $P_4$ and $P_5$. $P_4$ is a L-shaped plate with a cusp at the angle and is connected to $P_3$ at the dorsal end. $P_5$ is a rod that is connected to $P_4$ near the root of the cusp (Figure 4).

We assumed that the motion is symmetric with respect to the midline. Therefore the motion of the medial tooth was two-dimensional and only the left one of the

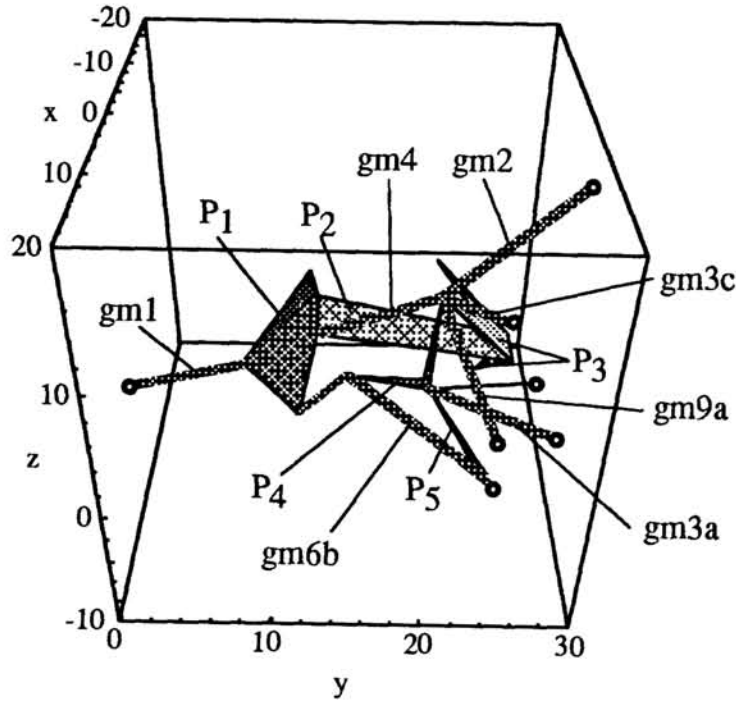

Figure 4: The design of the gastric mill model. Ossicle $P_1$ stands for the ossicles I and II, $P_2$ for VII, $P_3$ for VI, $P_4$ for III, IV, and V, $P_5$ for XIV in the standard description by Maynard and Dando (1974).

two lateral teeth was considered. The coordinate system was taken so that $x$-axis points to the left, $y$-axis backward, and $z$-axis upward. The rotation angles of the ossicles around $x$, $y$, and $z$ axes ware represented as $\theta$, $\phi$, and $\psi$ respectively. The configuration of the ossicles was determined by a 10 dimensional vector

$$\Theta = (y_0, z_0, \theta_1, \theta_2, \theta_3, \theta_4, \phi_4, \psi_4, \theta_5, \phi_5), \tag{5}$$

where $(y_0, z_0)$ represents the position of the joint between $P_1$ and $P_2$ and $(\theta_1, \theta_2, \theta_3)$ represents the rotation angle of $P_1$, $P_2$ and $P_3$ in the $y$-$z$ plane. The rotation angles of $P_4$ and $P_5$ were represented as $(\theta_4, \phi_4, \psi_4)$ and $(\theta_5, \phi_5)$ respectively. $P_5$ has only two degrees of rotation freedom since it is regarded as a rod.

We employed a quasi-static approximation. The configuration of the ossicles $\Theta$ was determined by the static balance of force. Now let $L_m$ and $F_m$ be the vectors of the muscle lengths and forces. Then the balance of the generalized forces in the $\Theta$ space (force for translation and torque for rotation) is given by

$$T_m(\Theta, F_m) + T_e = 0, \tag{6}$$

where $T_m$ and $T_e$ represent the generalized forces from muscles and external loads. The muscle force in the $\Theta$ space is given by

$$T_m(\Theta, F_m) = J(\Theta)^T F_m, \tag{7}$$

where $J(\Theta) = \partial L_m / \partial \Theta$ is the Jacobian matrix of the mapping $\Theta \mapsto L_m$ determined by the ossicle kinematics and the muscle attachment. Since it is very difficult to obtain a closed form solution of (6), we used a gradient descent equation

$$\frac{d\Theta}{dt} = -\varepsilon(T_m(\Theta, F_m) + T_e) = -\varepsilon(J(\Theta)^T F_m + T_e) \tag{8}$$

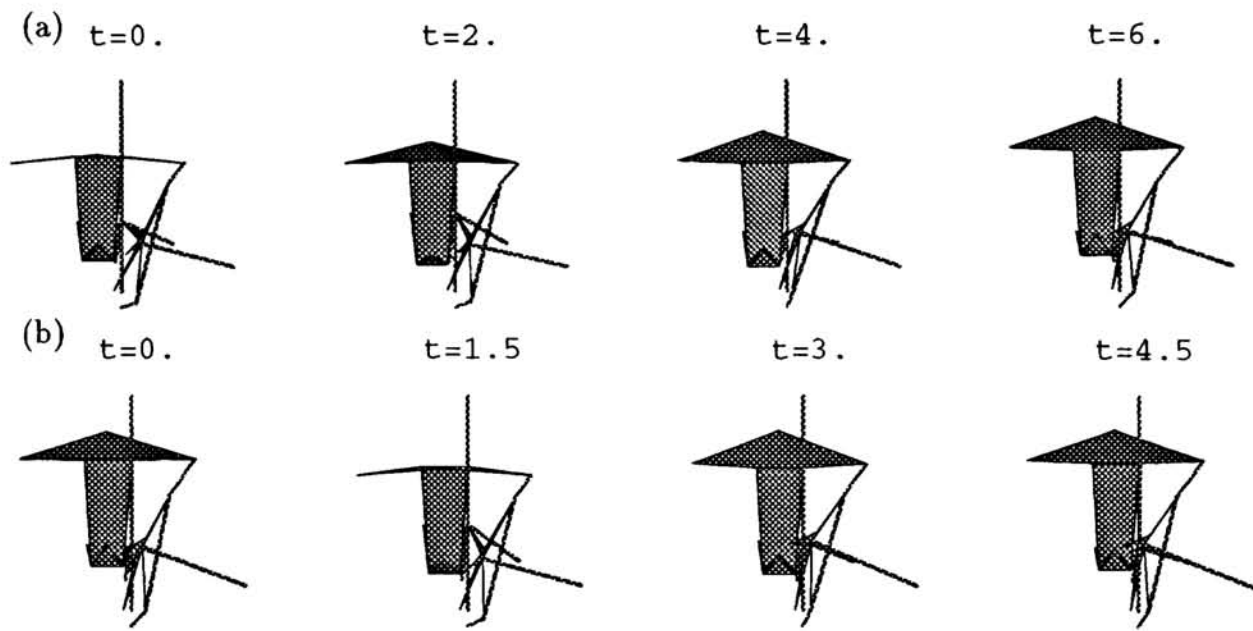

Figure 5: Chewing patterns predicted from oscillation patterns of isolated STG. (a) spontaneous pattern. (b) proctolin induced pattern.

to find the approximate solution of $\Theta(t)$. This is equivalent to assuming a viscosity term $\varepsilon^{-1}d\Theta/dt$ in the motion equation.

## 4    SIMULATION RESULTS

The musculoskeletal model is a 17-th order differential equation system and was integrated by Runge-Kutta method with a time step 1ms. Figure 5 shows examples of motion patterns predicted by the model. The motoneuron output of spontaneous oscillation of the isolated ganglion was used in (a) and the output under the effect of proctolin was used in (b). It has been reported in previous behavioral studies (Heinzel 1988b) that the dose of proctolin typically evokes "cut and grind" chewing pattern. The trajectory (b) predicted from the proctolin induced rhythm has a larger forward movement of the medial tooth while the lateral teeth are closed, which qualitatively agrees with the behavioral data.

## 5    DISCUSSION

The motor pattern generated by the model is considerably different from the chewing patterns observed in the intact animal using an endoscope. This is partly because of crude assumptions in model construction and errors in parameter estimation. However, this difference may also be due to the lack of sensory feedback in the isolated preparation. The future subject of this project is to refine the model so that we can reliably predict the motion from the neural outputs and to combine it with models of the gastric network (Rowat and Selverston, submitted) and sensory receptors. This will enable us to study how a biological control system integrates central pattern generation and sensory feedback.

## Acknowledgements

We thank Mike Beauchamp for the gm1 muscle data. This work was supported by the grant from Office of Naval Research N00014-91-J-1720.

## References

Boyle, M. E. T., Turrigiano, G. G., and Selverston, A. I. 1990. An endoscopic analysis of gastric mill movements produced by the peptide cholecystokinin. *Society for Neuroscience Abstracts* **16**, 724.

Elson, R. C. and Selverston, A. I. 1992. Mechanisms of gastric rhythm generation in the isolated stomatogastric ganglion of spiny lobsters: Bursting pacemaker potentials, synaptic interactions and muscarinic modulation. *Journal of Neurophysiology* **68**, 890–907.

Govind, C. K. and Lingle, C. J. 1987. Neuromuscular organization and pharmacology. In Selverston, A. I. and Moulins, M., editors, *The Crustacean Stomatogastric System*, pages 31–48. Springer-Verlag, Berlin.

Harris-Warrick, R. M., Marder, E., Selverston, A. I., and Moulins, M. 1992. *Dynamic Biological Networks — The Stomatogastric Nervous System*. MIT Press, Cambridge, MA.

Heinzel, H. G. 1988. Gastric mill activity in the lobster. I: Spontaneous modes of chewing. *Journal of Neurophysiology* **59**, 528–550.

Heinzel, H. G. 1988. Gastric mill activity in the lobster. II: Proctolin and octopamine initiate and modulate chewing. *Journal of Neurophysiology* **59**, 551–565.

Heinzel, H. G. and Selverston, A. I. 1988. Gastric mill activity in the lobster. III: Effects of proctolin on the isolated central pattern generator. *Journal of Neurophysiology* **59**, 566–585.

Hill, A. V. 1938. The heat of shortening and the dynamic constants of muscle. *Proceedings of the Royal Sciety of London, Series B* **126**, 136–195.

Hogan, N. 1984. Adaptive control of mechanical impedance by coactivation of antagonist muscles. *IEEE Transactions on Automatic Control* **29**, 681–690.

Maynard, D. M. and Dando, M. R. 1974. The structure of the stomatogastric neuromuscular system in callinectes sapidus, homarus americanus and panulirus argus (decapoda crustacea). *Philosophical Transactions of Royal Society of London, Biology* **268**, 161–220.

Rowat, P. F. and Selverston, A. I. Modeling the gastric mill central pattern generator of the lobster with a relaxation-oscillator network. submitted.

Selverston, A. I. and Moulins, M. 1987. *The Crustacean Stomatogastric System*. Springer-Verlag, New York, NY.

Turrigiano, G. G. and Selverston, A. I. 1990. A cholecystokinin-like hormone activates a feeding-related neural circuit in lobster. *Nature* **344**, 866–868.

Winters, J. M. 1990. Hill-based muscle models: A systems engineering perspective. In Winters, J. M. and Woo, S. L.-Y., editors, *Multiplie Muscle Systems: Biomechanics and Movement Organization*, chapter 5, pages 69–93. Springer-Verlag, New York, NY.

Zahalak, G. I. 1990. Modeling muscle mechanics (and energetics). In Winters, J. M. and Woo, S. L.-Y., editors, *Multiplie Muscle Systems: Biomechanics and Movement Organization*, chapter 1, pages 1–23. Springer-Verlag, New York, NY.
